# Hoeffding Races: Accelerating Model Selection Search for Classification and Function Approximation

**Oded Maron**
Artificial Intelligence Laboratory
Massachusetts Institute of Technology
Cambridge, MA 02139

**Andrew W. Moore**
Robotics Institute
School of Computer Science
Carnegie Mellon University
Pittsburgh, PA 15213

## Abstract

Selecting a good model of a set of input points by cross validation is a computationally intensive process, especially if the number of possible models or the number of training points is high. Techniques such as gradient descent are helpful in searching through the space of models, but problems such as local minima, and more importantly, lack of a distance metric between various models reduce the applicability of these search methods. Hoeffding Races is a technique for finding a good model for the data by quickly discarding bad models, and concentrating the computational effort at differentiating between the better ones. This paper focuses on the special case of leave-one-out cross validation applied to memory-based learning algorithms, but we also argue that it is applicable to any class of model selection problems.

## 1   Introduction

Model selection addresses "high level" decisions about how best to tune learning algorithm architectures for particular tasks. Such decisions include which function approximator to use, how to trade smoothness for goodness of fit and which features are relevant. The problem of *automatically* selecting a good model has been variously described as fitting a curve, learning a function, or trying to predict future

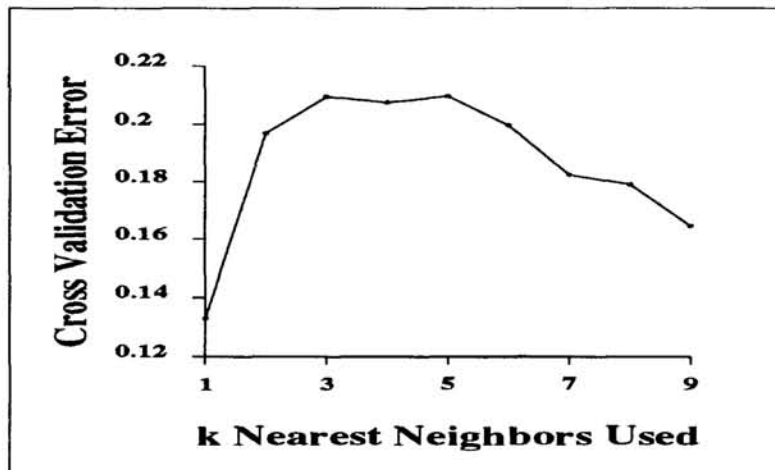

Figure 1: A space of models consisting of local-weighted-regression models with different numbers of nearest neighbors used. The global minimum is at one-nearest-neighbor, but a gradient descent algorithm would get stuck in local minima unless it happened to start in in a model where $k < 4$.

instances of the problem. One can think of this as a search through the space of possible models with some criterion of "goodness" such as prediction accuracy, complexity of the model, or smoothness. In this paper, this criterion will be prediction accuracy. Let us examine two common ways of measuring accuracy : using a test set and leave-one-out cross validation (Wahba and Wold, 1975).

- The **test set method** arbitrarily divides the data into a training set and a test set. The learner is trained on the training set, and is then queried with just the input vectors of the test set. The error for a particular point is the difference between the learner's prediction and the actual output vector.

- **Leave-one-out cross validation** trains the learner $N$ times (where $N$ is the number of points), each time omitting a different point. We attempt to predict each omitted point. The error for a particular point is the difference between the learner's prediction and the actual output vector.

The total error of either method is computed by averaging all the error instances.

The obvious method of searching through a space of models, the brute force approach, finds the accuracy of every model and picks the best one. The time to find the accuracy (error rate) of a particular model is proportional to the size of the test set $|TEST|$, or the size of the training set in the case of cross validation. Suppose that the model space is discretized into a finite number of models $|MODELS|$ — then the amount of work required is $O(|MODELS| \times |TEST|)$, which is expensive.

A popular way of dealing with this problem is gradient descent. This method can be applied to find the parameters (or weights) of a model. However, it cannot be used to find the structure (or architecture) of the model. There are two reasons for

this. First, we have empirically noted many occasions on which the search space is peppered with local minima (Figure 1). Second, at the highest level we are selecting from a set of entirely distinct models, with no numeric parameters over which to hill-climb. For example, is a neural net with 100 hidden units closer to a neural net with 50 hiden units or to a memory-based model which uses 3 nearest neighbors? There is no viable answer to this question since we cannot impose a viable metric on this model space.

The algorithm we describe in this paper, Hoeffding Races, combines the robustness of brute force and the computational feasibility of hill climbing. We instantiated the algorithm by specifying the set of models to be memory-based algorithms (Stanfill and Waltz, 1986) (Atkeson and Reinkensmeyer, 1989) (Moore, 1992) and the method of finding the error to be leave-one-out cross validation. We will discuss how to extend the algorithm to any set of models and to the test set method in the full paper. We chose memory-based algorithms since they go hand in hand with cross validation. Training is very cheap - simply keep all the points in memory, and all the algorithms of the various models can use the same memory. Finding the leave-one-out cross validation error at a point is cheap as making a prediction: simply "cover up" that point in memory, then predict its value using the current model. For a discussion of how to generate various memory-based models, see (Moore et al., 1992).

## 2   Hoeffding Races

The algorithm was inspired by ideas from (Haussler, 1992) and (Kaelbling, 1990) and a similar idea appears in (Greiner and Jurisica, 1992). It derives its name from Hoeffding's formula (Hoeffding, 1963), which concerns our confidence in the sample mean of $n$ independently drawn points $x_1, ..., x_n$. The probability of the estimated mean $E_{est} = \frac{1}{n} \sum_{1 \leq i \leq n} x_i$ being more than epsilon far away from the true mean $E_{true}$ after $n$ independently drawn points is bounded by:

$$Pr(|E_{true} - E_{est}| > \epsilon) < 2e^{-2n\epsilon^2/B^2}$$

where $B$ bounds the possible spread of point values.

We would like to say that with confidence $1 - \delta$, our estimate of the mean is within $\epsilon$ of the true mean; or in other words, $Pr(|E_{true} - E_{est}| > \epsilon) < \delta$. Combining the two equations and solving for $\epsilon$ gives us a bound on how close the estimated mean is to the true mean after $n$ points with confidence $1 - \delta$:

$$\epsilon = \sqrt{\frac{B^2 \log(2/\delta)}{2n}}$$

The algorithm starts with a collection of learning boxes. We call each model a learning box since we are treating the models as if they were black boxes. We are not looking at how complex or time-consuming each prediction is, just at the input and output of the box. Associated with each learning box are two pieces of information: a current estimate of its error rate and the number of points it has been tested upon so far. The algorithm also starts with a test set of size $N$. For leave-one-out cross validation, the test set is simply the training set.

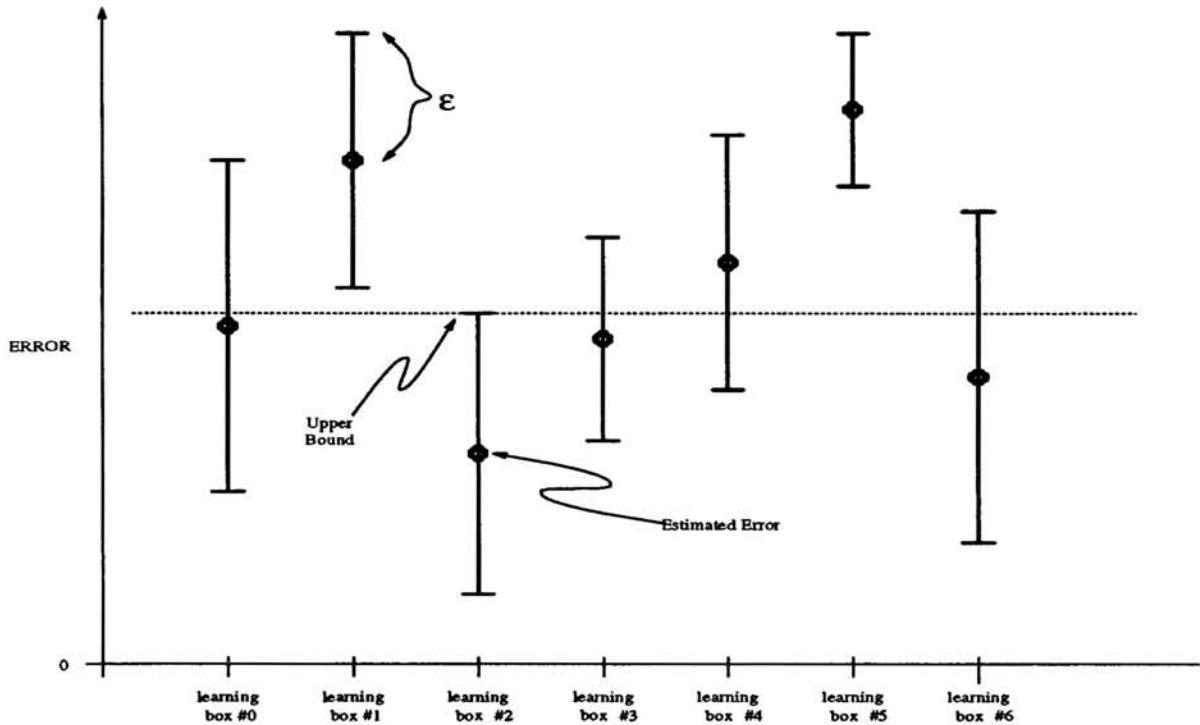

Figure 2: An example where the best upper bound of learning box #2 eliminates learning boxes #1 and #5. The size of $\epsilon$ varies since each learning box has its own upper bound on its error range, $B$.

At each point in the algorithm, we randomly select a point from the test set. We compute the error at that point for all learning boxes, and update each learning box's estimate of its own total error rate. In addition, we use Hoeffding's bound to calculate how close the current estimate is to the true error for each learning box. We then eliminate those learning boxes whose best possible error (their lower bound) is still greater than the worst error of the best learning box (its upper bound); see Figure 2. The intervals get smaller as more points are tested, thereby "racing" the good learning boxes, and eliminating the bad ones.

We repeat the algorithm until we are left with just one learning box, or until we run out of points. The algorithm can also be stopped once $\epsilon$ has reached a certain threshhold. The algorithm returns a set of learning boxes whose error rates are insignificantly (to within $\epsilon$) different after $N$ test points.

## 3    Proof of Correctness

The careful reader would have noticed that the confidence $\delta$ given in the previous section is incorrect. In order to prove that the algorithm indeed returns a set of learning boxes which includes the best one, we'll need a more rigorous approach. We denote by $\Delta$ the probability that the algorithm eliminates what would have been the best learning box. The difference between $\Delta$ and $\delta$ which was glossed over in the previous section is that $1 - \Delta$ is the confidence for the success of the entire algrithm, while $1 - \delta$ is the confidence in Hoeffding's bound for *one* learning box

during *one* iteration of the algorithm.

We would like to make a formal connection between $\Delta$ and $\delta$. In order to do that, let us make the requirement of a correct algorithm more stringent. We'll say that the algorithm is correct if every learning box is within $\epsilon$ of its true error at every iteration of the algorithm. This requirement encompasses the weaker requirement that we don't eliminate the best learning box. An algorithm is correct with confidence $\Delta$ if $Pr\{$ all learning boxes are within $\epsilon$ on all iterations$\} \geq 1 - \Delta$.

We'll now derive the relationship between $\delta$ and $\Delta$ by using the disjunctive probability inequality which states that $Pr\{A \vee B\} \leq Pr\{A\} + Pr\{B\}$.

Let's assume that we have $n$ iterations (we have $n$ points in our test set), and that we have $m$ learning boxes ($LB_1 \cdots LB_m$). By Hoeffding's inequality, we know that

$$Pr\{ \text{ a particular LB is within } \epsilon \text{ on a particular iteration}\} \geq 1 - \delta$$

Flipping that around we get:

$$Pr\{\text{a particular LB is wrong on a particular iteration}\} < \delta$$

Using the disjunctive inequality we can say

$$Pr\{ \quad \begin{aligned} &\textit{a particular LB is} \quad &&\textit{wrong on iteration 1 } \vee \\ &\textit{a particular LB is} \quad &&\textit{wrong on iteration 2 } \vee \\ &&&\cdots \\ &\textit{a particular LB is} \quad &&\textit{wrong on iteration n}\} \leq \delta \cdot n \end{aligned}$$

Let's rewrite this as:

$$Pr\{ \text{ a particular LB is wrong on any iteration}\} \leq \delta \cdot n$$

Now we do the same thing for all learning boxes:

$$Pr\{ \quad \begin{aligned} &LB_1 \textit{ is wrong on} \quad &&\textit{any iteration } \vee \\ &LB_2 \textit{ is wrong on} \quad &&\textit{any iteration } \vee \\ &&&\cdots \\ &LB_m \textit{ is wrong on} \quad &&\textit{any iteration}\} \leq \delta \cdot n \cdot m \end{aligned}$$

or in other words:

$$Pr\{ \text{ some LB is wrong in some iteration}\} \leq \delta \cdot n \cdot m$$

We flip this to get:

$$Pr\{\text{all LBs are within } \epsilon \text{ on all iterations}\} \geq 1 - \delta \cdot n \cdot m$$

Which is exactly what we meant by a correct algorithm with some confidence. Therefore, $\delta = \frac{\Delta}{n \cdot m}$. When we plug this into our expression for $\epsilon$ from the previous section, we find that we have only increased it by a constant factor. In other words, by pumping up $\epsilon$, we have managed to ensure the correctness of this algorithm with confidence $\Delta$. The new $\epsilon$ is expressed as:

$$\epsilon = \sqrt{\frac{B^2(\log(2nm) - \log(\Delta))}{n}}$$

Table 1: Test problems

| Problem | Description |
|---------|-------------|
| ROBOT | 10 input attributes, 5 outputs. Given an initial and a final description of a robot arm, learn the control needed in order to make the robot perform devil-sticking (Schaal and Atkeson, 1993). |
| PROTEIN | 3 inputs, output is a classification into one of three classes. This is the famous protein secondary structure database, with some preprocessing (Zhang et al., 1992). |
| ENERGY | Given solar radiation sensing, predict the cooling load for a building. This is taken from the Building Energy Predictor Shootout. |
| POWER | Market data for electricity generation pricing period class for the new United Kingdom Power Market. |
| POOL | The visually perceived mapping from pool table configurations to shot outcome for two-ball collisions (Moore, 1992). |
| DISCONT | An artificially constructed set of points with many discontinuities. Local models should outperform global ones. |

Clearly this is an extremely pessimistic bound and tighter proofs are possible (Omohundro, 1993).

## 4   Results

We ran Hoeffding Races on a wide variety of learning and prediction problems. Table 1 describes the problems, and Table 2 summarizes the results and compares them to brute force search.

For Table 2, all of the experiments were run using $\Delta = .01$. The initial set of possible models was constructed from various memory based algorithms: combinations of different numbers of nearest neighbors, different smoothing kernels, and locally constant vs. locally weighted regression. We compare the algorithms relative to the number of queries made, where a query is one learning box finding its error at one point. The brute force method makes $|TEST| \times |\text{LEARNING BOXES}|$ queries. Hoeffding Races eliminates bad learning boxes quickly, so it should make fewer queries.

## 5   Discussion

Hoeffding Races never does worse than brute force. It is least effective when all models perform equally well. For example, in the POOL problem, where there were 75 learning boxes left at the end of the race, the number of queries is only slightly smaller for Hoeffding Races than for brute force. In the ROBOT problem, where there were only 6 learning boxes left, a significant reduction in the number of queries can be seen. Therefore, Hoeffding Races is most effective when there exists a subset of clear winners within the initial set of models. We can then search over a very broad set of models without much concern about the computational expense

Table 2: Results of Brute Force vs. Hoeffding Races.

| Problem | points | Initial # learning boxes | queries with Brute Force | queries with Hoeffding Races | learning boxes left |
|---------|--------|--------------------------|--------------------------|------------------------------|---------------------|
| ROBOT   | 972    | 95   | 92340  | 15637  | 6  |
| PROTEIN | 4965   | 95   | 471675 | 349405 | 60 |
| ENERGY  | 2444   | 189  | 461916 | 121400 | 40 |
| POWER   | 210    | 95   | 19950  | 13119  | 48 |
| POOL    | 259    | 95   | 24605  | 22095  | 75 |
| DISCONT | 500    | 95   | 47500  | 25144  | 29 |

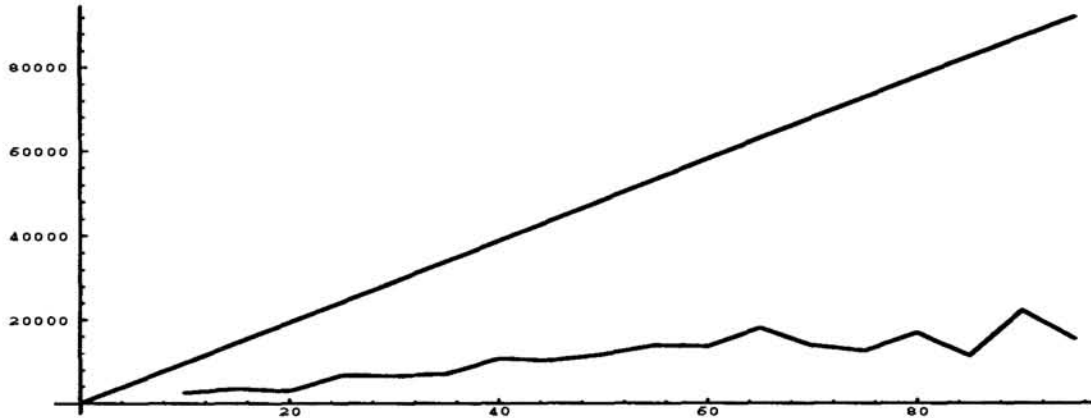

Figure 3: The x-axis is the size of a set of initial learning boxes (chosen randomly) and the y-axis is the number of queries to find a good model for the ROBOT problem. The bottom line shows performance by the Hoeffding Race algorithm, and the top line by brute force.

of a large initial set. Figure 3 demonstrates this. In all the cases we have tested, the learning box chosen by brute force is also contained by the set returned from Hoeffding Races. Therefore, there is no loss of performance accuracy.

The results described here show the performance improvement with relatively small problems. Preliminary results indicate that performance improvements will increase as the problems scale up. In other words, as the number of test points and the number of learning boxes increase, the ratio of the number of queries made by brute force to the number of queries made by Hoeffding Races becomes larger. However, the cost of each query then becomes the main computational expense.

## Acknowledgements

Thanks go to Chris Atkeson, Marina Meila, Greg Galperin, Holly Yanco, and Stephen Omohundro for helpful and stimulating discussions.

## References

[Atkeson and Reinkensmeyer, 1989] C. G. Atkeson and D. J. Reinkensmeyer. Using associative content-addressable memories to control robots. In W. T. Miller, R. S. Sutton, and P. J. Werbos, editors, *Neural Networks for Control*. MIT Press, 1989.

[Greiner and Jurisica, 1992] R. Greiner and I. Jurisica. A statistical approach to solving the EBL utility problem. In *Proceedings of the Tenth International conference on Artificial Intelligence (AAAI-92)*. MIT Press, 1992.

[Haussler, 1992] D. Haussler. Decision theoretic generalizations of the pac model for neural net and other learning applications. *Information and Computation*, 100:78–150, 1992.

[Hoeffding, 1963] Wassily Hoeffding. Probability inequalities for sums of bounded random variables. *Journal of the American Statistical Association*, 58:13–30, 1963.

[Kaelbling, 1990] L. P. Kaelbling. Learning in Embedded Systems. PhD. Thesis; Technical Report No. TR-90-04, Stanford University, Department of Computer Science, June 1990.

[Moore et al., 1992] A. W. Moore, D. J. Hill, and M. P. Johnson. An empirical investigation of brute force to choose features, smoothers and function approximators. In S. Hanson, S. Judd, and T. Petsche, editors, *Computational Learning Theory and Natural Learning Systems, Volume 3*. MIT Press, 1992.

[Moore, 1992] A. W. Moore. Fast, robust adaptive control by learning only forward models. In J. E. Moody, S. J. Hanson, and R. P. Lippman, editors, *Advances in Neural Information Processing Systems 4*. Morgan Kaufmann, April 1992.

[Omohundro, 1993] Stephen Omohundro. Private communication, 1993.

[Pollard, 1984] David Pollard. *Convergence of Stochastic Processes*. Springer-Verlag, 1984.

[Schaal and Atkeson, 1993] S. Schaal and C. G. Atkeson. Open loop stable control strategies for robot juggling. In *Proceedings of IEEE conference on Robotics and Automation*, May 1993.

[Stanfill and Waltz, 1986] C. Stanfill and D. Waltz. Towards memory-based reasoning. *Communications of the ACM*, 29(12):1213–1228, December 1986.

[Wahba and Wold, 1975] G. Wahba and S. Wold. A completely automatic french curve: Fitting spline functions by cross-validation. *Communications in Statistics*, 4(1), 1975.

[Zhang et al., 1992] X. Zhang, J.P. Mesirov, and D.L. Waltz. Hybrid system for protein secondary structure prediction. *Journal of Molecular Biology*, 225:1049–1063, 1992.